# Classifying Single Trial EEG: Towards Brain Computer Interfacing

**Benjamin Blankertz**[1]*, **Gabriel Curio**[2] **and Klaus-Robert Müller**[1,3]

[1]Fraunhofer-FIRST.IDA, Kekuléstr. 7, 12489 Berlin, Germany

[2]Neurophysics Group, Dept. of Neurology, Klinikum Benjamin Franklin, Freie Universität Berlin, Hindenburgdamm 30, 12203 Berlin, Germany

[3]University of Potsdam, Am Neuen Palais 10, 14469 Potsdam, Germany

{benjamin.blankertz,klaus-robert.mueller}@first.fraunhofer.de,
curio@zedat.fu-berlin.de

## Abstract

Driven by the progress in the field of single-trial analysis of EEG, there is a growing interest in brain computer interfaces (BCIs), i.e., systems that enable human subjects to control a computer only by means of their brain signals. In a pseudo-online simulation our BCI detects upcoming finger movements in a natural keyboard typing condition and predicts their laterality. This can be done on average 100–230 ms *before* the respective key is actually pressed, i.e., long before the onset of EMG. Our approach is appealing for its short response time and high classification accuracy (>96%) in a binary decision where no human training is involved. We compare discriminative classifiers like Support Vector Machines (SVMs) and different variants of Fisher Discriminant that possess favorable regularization properties for dealing with high noise cases (inter-trial variablity).

## 1 Introduction

The online analysis of single-trial electroencephalogram (EEG) measurements is a challenge for signal processing and machine learning. Once the high inter-trial variability (see Figure 1) of this complex multivariate signal can be reliably processed, the next logical step is to make use of the brain activities for real-time control of, e.g., a computer. In this work we study a *pseudo-online* evaluation of single-trial EEGs from voluntary self-paced finger movements and exploit the laterality of the left/right hand signal as one bit of information for later control. Features of our BCI approach are (a) *no pre-selection* for artifact trials, (b) state-of-the-art learning machines with inbuilt feature selection mechanisms (i.e., sparse Fisher Discriminant Analysis and SVMs) that lead to >96% classification accuracies, (c) *non-trained users* and (d) short response times. Although our setup was not tuned for speed, the *a posteriori* determined information transmission rate is 23 bits/min which makes our approach competitive to existing ones (e.g., [1, 2, 3, 4, 5, 6, 7]) that will be discussed in section 2.

**Aims and physiological concept of BCI devices.** Two key issues to start with when conceiving a BCI are (1) the definition of a behavioral context in which a subject's brain signals will be monitored and used eventually as surrogate for a bodily, e.g., manual, input of computer commands, and (2) the choice of brain signals which are optimally capable to convey the subject's intention to the computer.

Concerning the behavioral context, typewriting on a computer keyboard is a highly over-learned motor competence. Accordingly, a natural first choice is a BCI-situation which induces the subject to arrive at a particular decision that is coupled to a predefined (learned) motor output. This approach is well known as a two alternative forced choice-reaction task (2AFC) where one out of two stimuli (visual, auditory or somatosensory) has to be detected, categorised and responded to by issuing one out of two alternative motor commands, e.g., pushing a button with either the left or right hand. A task variant without explicit sensory input is the voluntary, *endogeneous* generation of a ›go‹ command involving the deliberate choice between the two possible motor outputs at a self-paced rate. Here, we chose this latter approach so as to approximate the natural computer input situation of *self-paced typewriting*.

Concerning the selection of brain signals related to such endogeneous motor commands we focussed here on one variant of *slow* brain potentials which are specifically related to the preparation and execution of a motor command, rather than reflecting merely unspecific modulations of vigilance or attention. Using multi-channel EEG-mapping it has been repeatedly demonstrated that several highly localised brain areas contribute to cerebral motor command processes. Specifically, a negative ›Bereitschaftspotential‹ (BP) precedes the voluntary initiation of the movement. A differential scalp potential distribution can be reliably demonstrated in a majority of experimental subjects with larger BP at lateral scalp positions (C3, C4) positioned over the left or right hemispherical primary motor cortex, respectively, consistenly correlating with the performing (right or left) hand [8, 9].

Because one potential BCI-application is with paralysed patients, one might consider to mimic the ›no-motor-output‹ of these individuals by having healthy experimental subjects to intend a movement but to withhold its execution (*motor imagery*). While it is true that brain potentials comparable to BP are associated with an imagination of hand movements, which indeed is consistent with the assumption that the primary motor cortex is active with motor imagery, actual motor performance significantly increased these potentials [10]. We therefore chose to instruct the experimental subjects to actually perform the typewriting finger movements, rather than to merely imagine their performance, for two reasons: first, this will increase the BP signal strength optimising the signal-to-noise ratio in BCI-related single trial analyses; and second, we propose that it is important for the subject's task efficiency not to be engaged in an unnatural condition where, in addition to the preparation of a motor command, a second task, i.e., to ›veto‹ the very same movement, has to be executed. In the following section we will briefly review part of the impressive earlier research towards BCI devices (e.g., [1, 2, 3, 4, 5, 6, 7]) before experimental set-up and classification results are discussed in sections 3 and 4 respectively. Finally a brief conclusion in given.

## 2   A brief outline of BCI research

Birbaumer et al. investigate slow cortical potentials (SCP) and how they can be self-regulated in a feedback scenario. In their thought translation device [2] patients learn to produce cortical negativity or positivity at a central scalp location at will, which is fed back to the user. After some *training* patients are able to transmit binary decisions in a 4 sec periodicity with accuracy levels up to 85% and therewith control a language support program or an internet browser.

Pfurtscheller et al. built a BCI system based on event-related (de-)synchronisation (ERD/ERS, typically of the $\mu$ and central $\beta$ rhythm) for online classification of movement imaginations or preparations into 2–4 classes (e.g., left/right index finger, feet, tongue).

Typical preprocessing techniques are adaptive autoregressive parameters, common spatial patterns (after band pass filtering) and band power in subject specific frequency bands. Classification is done by Fisher discriminant analysis, multi-layer neural networks or LVQ variants. In classification of exogeneous movement preparations, rates of 98%, 96% and 75% (for three subjects respectively) are obtained before movement onset[1] in a 3 classes task and trials of 8 sec [3]. Only *selected*, artifact free trials (less that 40%) were used. A tetraplegic patient controls his hand orthosis using the Graz BCI system.

Wolpaw et al. study EEG-based cursor control [4], translating the power in subject specific frequency bands, or autoregressive parameters, from two spatially filtered scalp locations over sensorimotor cortex into vertical cursor movement. Users initially gain control by various kinds of motor imagery (the setting favours ›movement‹ vs. ›no movement‹ in contrast to ›left‹ vs. ›right‹), which they report to use less and less as feedback training continues. In cursor control trials of at least 4 sec duration *trained* subjects reach accuracies of over 90%. Some subjects acquired also considerable control in a 2-d setup.

## 3  Acquisition and preprocessing of brain signals

**Experimental setup.**    The subject sat in a normal chair, relaxed arms resting on the table, fingers in the standard typing position at the computer keyboard. The task was to press with the index and little fingers the corresponding keys in a self-chosen order and timing (›self-paced key typing‹). The experiment consisted of 3 sessions of 6 minutes each, pre- and postceeded by 60 seconds relaxing phase. All sessions were conducted on the same day with some minutes break inbetween. Typing of a total of 516 keystrokes was done at an average speed of 1 key every 2.1 seconds.

Brain activity was measured with 27 Ag/AgCl electrodes at positions of the extended international 10-20 system, 21 mounted over motor and somatosensory cortex, 5 frontal and one occipital, referenced to nasion (sampled at 1000 Hz, band-pass filtered 0.05–200 Hz). Besides EEG we recorded an electromyogram (EMG) of the *musculus flexor digitorum* bilaterally (10–200 Hz) and a horizontal and vertical electrooculogram (EOG). In an event channel the timing of keystrokes was stored along with the EEG signal. All data were recorded with a NeuroScan device and converted to Matlab format for further analysis. The signals were downsampled to 100 Hz by picking every 10th sample. In a moderate rejection we sorted out only 3 out of 516 trials due to heavy measurement artifacts, while keeping trials that are contaminated by less serious artifacts or eye blinks. Note that 0.6% rejection rate is very low in contrast to most other BCI offline studies.

**The issue of preprocessing.**    Preprocessing the data can have a substantial effect on classification in terms of accuracy, effort and suitability of different algorithms. The question to what degree data should be preprocessed prior to classification is a trade-off between the danger of loosing information or overfitting and not having enough training samples for the classifier to generalize from high dimensional, noisy data. We have investigated two options: unprocessed data and preprocessing that was designed to focus on BP related to finger movement:

   (none)  take 200 ms of raw data of all relevant channels;
   ($<5$ Hz)  filter the signal low pass at 5 Hz, subsample it at 20 Hz and take 150 ms of all relevant channels (see Figure 1);

Speaking of classification at a certain time point we strictly mean classification based on EEG signals until that very time point. The following procedure of calculating features of a single trial due to ($<5$ Hz) is easy applicable in an online scenario: Take the last 128 sample points of each channel (to the past relative from the given time point), apply a windowed ($w(n) := 1 - \cos(n\pi/128)$) FFT, keep only the coefficients corresponding to the pass

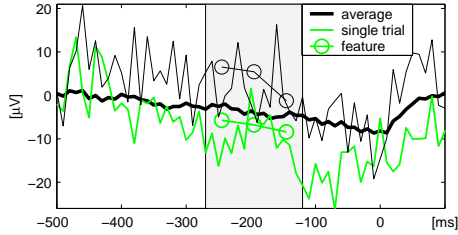

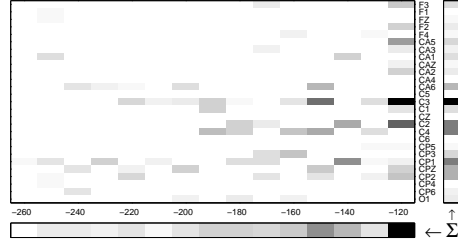

Figure 1: Averaged data and two single trials of right finger movements in channel C3. 3 values (marked by circles) of smoothed signals are taken as features in each channel.

Figure 2: Sparse Fisher Discriminant Analysis selected 68 features (shaded) from 405 input dimensions (27 channels × 15 samples [150 ms]) of raw EEG data.

band (bins 2–7, as bin 1 just contains DC information) and transform back. Downsampling to 20 Hz is done by calculating the mean of consecutive 5-tuple of data points. We investigated the alternatives of taking all 27 channels, or only the 21 located over motor and sensorimotor cortex. The 6 frontal and occipital channels are expected not to give strong contributions to the classification task. Hence a comparison shows, whether a classifier is disturbed by low information channels or if it even manages to extract information from them.

Figure 1 depicts two single trial EEG signals at scalp location C3 for right finger movements. These two single trials are very well-shaped and were selected for resembling the the grand average over all 241 right finger movements, which is drawn as thick line. Usually the BP of a single trial is much more obscured by non task-related brain activity and noise. The goal of preprocessing is to reveal task-related components to a degree that they can be detected by a classifier. Figure 1 shows also the feature vectors due to preprocessing ($<5$ Hz) calculated from the depicted raw single trial signals.

## 4    From response-aligned to online classification

We investigate some linear classification methods. Given a linear classifier $(w,b)$ in separating hyperplane formulation ($w^\top x + b = 0$), the estimated label $\{1, -1\}$ of an input vector $x \in \mathbb{R}^N$ is $\hat{y} = \text{sign}(w^\top x + b)$. If no a priori knowledge on the probability distribution of the data is available, a typical objective is to minimize a combination of empirical risk function and some regularization term that restrains the algorithm from overfitting to the training set $\{(x_k, y_k) \mid k = 1, \dots, K\}$. Taking a soft margin loss function [11] yields the empirical risk function $\sum_{k=1}^{K} \max(0, 1 - y_k(w^\top x_k + b))$. In most approaches of this type there is a hyper-parameter that determines the trade-off between risk and regularization, which has to be chosen by model selection on the training set[2].

**Fisher Discriminant (FD)** is a well known classification method, in which a projection vector is determined to maximize the distance between the projected means of the two classes while minimizing the variance of the projected data within each class [13]. In the binary decision case FD is equivalent to a least squares regression to (properly scaled) class labels.

**Regularized Fisher Discriminant (RFD)** can be obtained via a mathematical programming approach [14]:

$$\min_{w,b,\xi} \; 1/2 \|w\|_2^2 \; + \; C/K \|\xi\|_2^2 \quad \text{subject to}$$

$$y_k(w^\top x_k + b) = 1 - \xi_k \quad \text{for } k = 1, \dots, K$$

| filter | ch's | FD | RFD | SFD | SVM | $k$-NN |
|--------|------|-----|-----|-----|-----|--------|
| <5 Hz | mc | 3.7±2.6 | 3.3±2.2 | 3.3±2.2 | 3.2±2.5 | 21.6±4.9 |
| <5 Hz | all | 3.3±2.5 | 3.1±2.5 | 3.4±2.7 | 3.6±2.5 | 23.1±5.8 |
| none | mc | 18.1±4.8 | 7.0±4.1 | 6.4±3.4 | 8.5±4.3 | 29.6±5.9 |
| none | all | 29.3±6.1 | 7.5±3.8 | 7.0±3.9 | 9.8±4.4 | 32.2±6.8 |

Table 3: Test set error ($\pm$ std) for classification at 120 ms before keystroke; ›mc‹ refers to the 21 channels over (sensori) motor cortex, ›all‹ refers to all 27 channels.

where $\|\cdot\|_2$ denotes the $\ell_2$-norm ($\|w\|_2^2 = w^\top w$) and $C$ is a model parameter. The constraint $y_k(w^\top x_k + b) = 1 - \xi_k$ ensures that the class means are projected to the corresponding class labels, i.e., 1 and $-1$. Minimizing the length of $w$ maximizes the margin between the projected class means relative to the intra class variance. This formalization above gives the opportunity to consider some interesting variants, e.g.,

**Sparse Fisher Discriminant (SFD)** uses the $\ell_1$-norm ($\|w\|_1 = \Sigma|w_n|$) on the regularizer, i.e., the goal function is $1/N\|w\|_1 + C/K\|\xi\|_2^2$. This choice favours solutions with sparse vectors $w$, so that this method also yields some feature selection (in input space). When applied to our raw EEG signals SFD selects 68 out of 405 input dimensions that allow for a left vs. right classification with good generalization. The choice coincides in general with what we would expect from neurophysiology, i.e., high loadings for electrodes close to left and right hemisphere motor cortices which increase prior to the keystroke, cf. Figure 2. But here the selection is automatically adapted to subject, electrode placement, etc.

Our implementation of RFD and SFD uses the *cplex* optimizer [15].

**Support Vector Machines (SVMs)** are well known for their use with kernels [16, 17]. Here we only consider linear SVMs:

$$\min_{w,b,\xi} 1/2\|w\|_2^2 + C/K\|\xi\|_1 \quad \text{subject to}$$

$$y_k(w^\top x_k + b) \geqslant 1 - \xi_k, \quad \text{and} \quad \xi_k \geqslant 0$$

The choice of regulization keeps a bound on the Vapnik-Chervonenkis dimension small. In an equivalent formulation the objective is to maximize the margin between the two classes (while minimizing the soft margin loss function)[3].

For comparision we also employed a standard classifier of different type:

**k-Nearest-Neighbor (k-NN)** maps an input vector to that class to which the majority of the $k$ nearest training samples belong. Those neighbors are determined by euclidean distance of the corresponding feature vectors. The value of $k$ chosen by model selection was around 15 for processed and around 25 for unprocessed data.

**Classification of response-aligned windows.** In the first phase we make full use of the information that we have regarding the timing of the keystrokes. For each single trial we calculate a feature vector as described above with respect to a fixed timing relative to the key trigger (›response-aligned‹). Table 3 reports the mean error on test in a $10\times10$-fold crossvalidation for classifying in ›left‹ and ›right‹ at 120 ms prior to keypress. Figure 4 shows the time course of the classification error. For comparison, the result of EMG-based classification is also displayed. It is more or less at chance level up to 120 ms before the keystroke. After that the error rate decreases rapidly. Based upon this observation we chose $t = -120$ ms for investigating EEG-based classification. From Table 3 we see that FD works well with the preprocssed data, but as dimensionality increases the performance breaks down. $k$-NN is not successful at all. The reason for the failure is that the variance in the discriminating directions is much smaller that the variance in other directions. So using the euclidean metric is not an appropriate similarity measure for this purpose. All regularized discriminative classifiers attain comparable results. For preprocessed data a very low

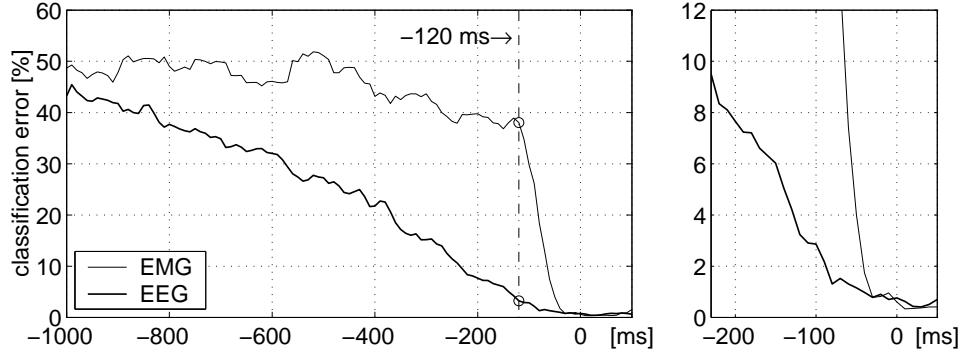

Figure 4: Comparison of EEG ($<5\,$Hz, mc, SFD) and EMG based classification with respect to the endpoint of the classification interval. The right panel gives a details view: -230 to 50 ms.

error rate between 3% and 4% can be reached without a significant difference between the competing methods. For the classification of raw data the error is roughly twice as high. The concept of seeking sparse solution vectors allows SFD to cope very well with the high dimensional raw data. Even though the error is twice as high compared to the the minimum error, this result is very interesting, because it does not rely on preprocessing. So the SFD approach may be highly useful for online situations, when no precursory experiments are available for tuning the preprocessing.

The comparison of EEG- and EMG-based classification in Figure 4 demonstrates the rapid response capability of our system: 230 ms before the actual keystroke the classification rate exceeds 90%. To assess this result it has to be recalled that movements were performed spontaneously. At $-120\,$ms, when the EMG derived classifier is still close to chance, EEG based classification becomes already very stable with less than 3.5% errors. Interpreting the last result in the sense of a 2AFC gives an information transfer rate of $^{60}/_{2.1}B \approx 22.9$ [bits/min], where $B = \log_2 N + p\log_2 p + (1-p)\log_2(^{1-p}/_{N-1})$ is the number of bits per selection from $N = 2$ choices with success probability $p = 1 - 0.035$ (under some uniformity assumptions).

**Classification in sliding windows.**     The second phase is an important step towards online classification of endogenous brain signals. We have to refrain from using event timing information (e.g., of keystrokes) in the test set. Accordingly, classification has to be performed in sliding windows and the classifier does not know in what time relation the given signals are to the event—maybe there is even no event. Technically classification could be done as before, as the trained classifiers can be applied to the feature vectors calculated from some arbitrary window. But in practice this is very likely to lead to unreliable results since those classifiers are highly specialized to signals that have a certain time relation to the response. The behavior of the classifier elsewhere is uncertain. The typical way to make classification more robust to time shifted signals is jittered training. In our case we used 4 windows for each trial, ending at -240, -160, -80 and 0 ms relative to the response (i.e., we get four feature vectors from each trial).

**Movement detection and pseudo-online classification.**     Detecting upcoming events is a crucial point in online analysis of brain signals in an unforced condition. To accomplish this, we employ a second classifier that distinguishes movement events from the ›rest‹. Figures 5 and 6 display the continuous classifier output $w^\top x + b$ (henceforth called graded) for left/right and movement/rest distinction, respectively. For Figure 5, a classifier was trained as described above and subsequently applied to windows sliding over unseen test samples yielding ›traces‹ of graded classifier outputs. After doing this for several train/test set splits, the borders of the shaded tubes are calculated as 5 and 95 percentile values of

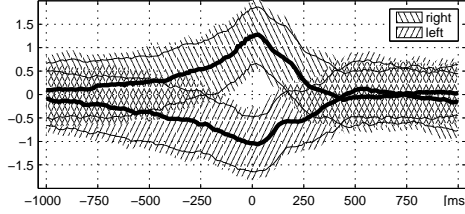

Figure 5: Graded classifier output for left/right distinctions.

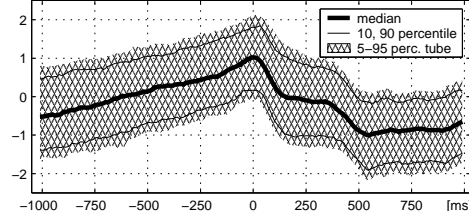

Figure 6: Graded classifier output for movement detection in endogenous brain signals.

those traces, thin lines are at 10 and 90 percent, and the thick line indicates the median. At $t = -100$ ms the median for right events in Figure 5 is approximately 0.9, i.e., applying the classifier to right events from the test set yielded in 50% of the cases an output greater 0.9 (and in 50% an output less than 0.9). The corresponding 10-percentile line is at 0.25 which means that the output to 90% of the right events was greater than 0.25. The second classifier (Figure 6) was trained for class ›movement‹ on all trials with jitters as described above and for class ›rest‹ in multiple windows between the keystrokes. The preprocessing and classification procedure was the same as for left vs. right.

The classifier in Figure 5 shows a pronounced separation during the movement (preparation and execution) period. In other regions there is an overlap or even crossover of the classifier outputs of the different classes. From Figure 5 we observe that the left/right classifier alone does not distinguish reliably between ›movement‹ and ›no movement‹ by the magnitude of its output, which explains the need for a movement detector. The elevation for the left class is a little less pronounced (e.g., the median is $-1$ at $t = 0$ ms compared to 1.2 for right events) which is probably due to the fact that the subject is right-handed. The movement detector in Figure 6 brings up the movement phase while giving (mainly) negative output to the post movement period. This differentiation is not as decisive as desirable, hence further research has to be pursued to improve on this. Nevertheless a pseudo-online BCI run on the recorded data using a combination of the two classifiers gave the very satisfying result of around 10% error rate. Taking this as a 3 classes choice (left, right, none) this corresponds to an information transmission rate of 29 bits/min.

## 5  Concluding discussion

We gave an outline of our BCI system in the experimental context of voluntary self-paced movements. Our approach has the potential for high bit rates, since (1) it works at a high trial frequency, and (2) classification errors are very low. So far we have used untrained individuals, i.e., improvement can come from appropriate training schemes to shape the brain signals. The two-stage process of first a meta classification whether a movement is about to take place and then a decision between left/right finger movement is very natural and an important new feature of the proposed system. Furthermore, we reject only 0.6% of the trials due to artifacts, so our approach seems ideally suited for the true, highly noisy feedback BCI scenario. Finally, the use of state-of-the-art learning machines enables us not only to achieve high decision accuracies, but also, as a by-product of the classification, the few most prominent features that are found match nicely with physiological intuition: the most salient information can be gained between 230–100 ms before the movement with a focus on C3/C4 area, i.e., over motor cortices, cf. Figure 2.

There are clear perspectives for improvement in this BCI approach: our future research activities will therefore focus on (a) projection techniques like ICA, (b) time-series approaches to capture the (non-linear) dynamics of the multivariate EEG signals, and (c) construction of specially adapted kernel functions (SVM or kernel FD) in the spirit of, e.g., [17] to ultimately obtain a BCI feedback system with an even higher bit rate and accuracy.

**Acknowledgements.**     We thank S. Harmeling, M. Kawanabe, J. Kohlmorgen, J. Laub, S. Mika, G. Rätsch, R. Vigário and A. Ziehe for helpful discussions.

## Footnotes

*To whom correspondence should be addressed.

[1] Precisely: before mean EMG onset time, for some trials this is before for others after EMG onset.

[2]As this would be very time consuming in *k*-fold crossvalidation, we proceed similarly to [12].

[3]We used the implementation LIBSVM by Chang and Lin, available along with other implementations from `www.kernel-machines.org`.

## References

[1] J. J. Vidal, "Toward direct brain-computer communication", *Annu. Rev. Biophys.*, 2: 157–180, 1973.

[2] N. Birbaumer, N. Ghanayim, T. Hinterberger, I. Iversen, B. Kotchoubey, A. Kübler, J. Perelmouter, E. Taub, and H. Flor, "A spelling device for the paralysed", *Nature*, 398: 297–298, 1999.

[3] B. O. Peters, G. Pfurtscheller, and H. Flyvbjerg, "Automatic Differentiation of Multichannel EEG Signals", *IEEE Trans. Biomed. Eng.*, 48(1): 111–116, 2001.

[4] J. R. Wolpaw, D. J. McFarland, and T. M. Vaughan, "Brain-Computer Interface Research at the Wadsworth Center", *IEEE Trans. Rehab. Eng.*, 8(2): 222–226, 2000.

[5] W. D. Penny, S. J. Roberts, E. A. Curran, and M. J. Stokes, "EEG-based cummunication: a pattern recognition approach", *IEEE Trans. Rehab. Eng.*, 8(2): 214–215, 2000.

[6] J. D. Bayliss and D. H. Ballard, "Recognizing Evoked Potentials in a Virtual Environment", in: S. A. Solla, T. K. Leen, and K.-R. Müller, eds., *Advances in Neural Information Processing Systems*, vol. 12, 3–9, MIT Press, 2000.

[7] S. Makeig, S. Enghoff, T.-P. Jung, and T. J. Sejnowski, "A Natural Basis for Efficient Brain-Actuated Control", *IEEE Trans. Rehab. Eng.*, 8(2): 208–211, 2000.

[8] W. Lang, O. Zilch, C. Koska, G. Lindinger, and L. Deecke, "Negative cortical DC shifts preceding and accompanying simple and complex sequential movements", *Exp. Brain Res.*, 74(1): 99–104, 1989.

[9] R. Q. Cui, D. Huter, W. Lang, and L. Deecke, "Neuroimage of voluntary movement: topography of the Bereitschaftspotential, a 64-channel DC current source density study", *Neuroimage*, 9(1): 124–134, 1999.

[10] R. Beisteiner, P. Hollinger, G. Lindinger, W. Lang, and A. Berthoz, "Mental representations of movements. Brain potentials associated with imagination of hand movements", *Electroencephalogr. Clin. Neurophysiol.*, 96(2): 183–193, 1995.

[11] K. P. Bennett and O. L. Mangasarian, "Robust Linear Programming Discrimination of two Linearly Inseparable Sets", *Optimization Methods and Software*, 1: 23–34, 1992.

[12] G. Rätsch, T. Onoda, and K.-R. Müller, "Soft Margins for AdaBoost", 42(3): 287–320, 2001.

[13] R. O. Duda, P. E. Hart, and D. G. Stork, *Pattern Classification*, Wiley & Sons, 2nd edn., 2001.

[14] S. Mika, G. Rätsch, and K.-R. Müller, "A mathematical programming approach to the Kernel Fisher algorithm", in: T. K. Leen, T. G. Dietterich, and V. Tresp, eds., *Advances in Neural Information Processing Systems 13*, 591–597, MIT Press, 2001.

[15] "ILOG Solver, ILOG CPLEX 6.5 Reference Manual", www.ilog.com, 1999.

[16] V. Vapnik, *The nature of statistical learning theory*, Springer Verlag, New York, 1995.

[17] K.-R. Müller, S. Mika, G. Rätsch, K. Tsuda, and B. Schölkopf, "An Introduction to Kernel-Based Learning Algorithms", *IEEE Transactions on Neural Networks*, 12(2): 181–201, 2001.